# Linearly-solvable Markov decision problems

**Emanuel Todorov**

Department of Cognitive Science
University of California San Diego
todorov@cogsci.ucsd.edu

## Abstract

We introduce a class of MPDs which greatly simplify Reinforcement Learning. They have discrete state spaces and continuous control spaces. The controls have the effect of rescaling the transition probabilities of an underlying Markov chain. A control cost penalizing KL divergence between controlled and uncontrolled transition probabilities makes the minimization problem convex, and allows analytical computation of the optimal controls given the optimal value function. An exponential transformation of the optimal value function makes the minimized Bellman equation linear. Apart from their theoretical significance, the new MDPs enable efficient approximations to traditional MDPs. Shortest path problems are approximated to arbitrary precision with largest eigenvalue problems, yielding an $O(n)$ algorithm. Accurate approximations to generic MDPs are obtained via continuous embedding reminiscent of LP relaxation in integer programming. Off-policy learning of the optimal value function is possible without need for state-action values; the new algorithm (Z-learning) outperforms Q-learning.

**This work was supported by NSF grant ECS–0524761**.

## 1   Introduction

In recent years many hard problems have been transformed into easier problems that can be solved efficiently via linear methods [1] or convex optimization [2]. One area where these trends have not yet had a significant impact is Reinforcement Learning. Indeed the discrete and unstructured nature of traditional MDPs seems incompatible with simplifying features such as linearity and convexity. This motivates the search for more tractable problem formulations. Here we construct the first MDP family where the minimization over the control space is convex and analytically tractable, and where the Bellman equation can be exactly transformed into a linear equation. The new formalism enables efficient numerical methods which could not previously be applied in Reinforcement Learning. It also yields accurate approximations to traditional MDPs.

Before introducing our new family of MDPs, we recall the standard formalism. Throughout the paper $\mathcal{S}$ is a finite set of states, $\mathcal{U}(i)$ is a set of admissible controls at state $i \in \mathcal{S}$, $\ell(i, u) \geq 0$ is a cost for being in state $i$ and choosing control $u \in \mathcal{U}(i)$, and $P(u)$ is a stochastic matrix whose element $p_{ij}(u)$ is the transition probability from state $i$ to state $j$ under control $u$. We focus on problems where a non-empty subset $\mathcal{A} \subseteq \mathcal{S}$ of states are absorbing and incur zero cost: $p_{ij}(u) = \delta_i^j$ and $\ell(i, u) = 0$ whenever $i \in \mathcal{A}$. Results for other formulations will be summarized later. If $\mathcal{A}$ can be reached with non-zero probability in a finite number of steps from any state, then the undiscounted infinite-horizon optimal value function is finite and is the unique solution [3] to the Bellman equation

$$v(i) = \min_{u \in \mathcal{U}(i)} \left\{ \ell(i, u) + \sum_j p_{ij}(u) v(j) \right\} \tag{1}$$

For generic MDPs this equation is about as far as one can get analytically.

## 2 A class of more tractable MDPs

In our new class of MDPs the control $\mathbf{u} \in \mathbb{R}^{|\mathcal{S}|}$ is a real-valued vector with dimensionality equal to the number of discrete states. The elements $u_j$ of $\mathbf{u}$ have the effect of directly modifying the transition probabilities of an uncontrolled Markov chain. In particular, given an uncontrolled transition probability matrix $\overline{P}$ with elements $\overline{p}_{ij}$, we define the controlled transition probabilities as

$$p_{ij}(\mathbf{u}) = \overline{p}_{ij} \exp(u_j) \tag{2}$$

Note that $P(0) = \overline{P}$. In some sense this is the most general notion of "control" one can imagine – we are allowing the controller to rescale the underlying transition probabilities in any way it wishes. However there are two constraints implicit in (2). First, $\overline{p}_{ij} = 0$ implies $p_{ij}(\mathbf{u}) = 0$. In this case $u_j$ has no effect and so we set it to $0$ for concreteness. Second, $P(\mathbf{u})$ must have row-sums equal to $1$. Thus the admissible controls are

$$\mathcal{U}(i) = \left\{ \mathbf{u} \in \mathbb{R}^{|\mathcal{S}|}; \sum_j \overline{p}_{ij} \exp(u_j) = 1; \ \overline{p}_{ij} = 0 \implies u_j = 0 \right\} \tag{3}$$

Real-valued controls make it possible to define a natural control cost. Since the control vector acts directly on the transition probabilities, it makes sense to measure its magnitude in terms of the difference between the controlled and uncontrolled transition probabilities. Differences between probability distributions are most naturally measured using KL divergence, suggesting the following definition. Let $\mathbf{p}_i(\mathbf{u})$ denote the $i$-th row-vector of the matrix $P(\mathbf{u})$, that is, the vector of transition probabilities from state $i$ to all other states under control $\mathbf{u}$. The control cost is defined as

$$r(i, \mathbf{u}) = \mathrm{KL}\left(\mathbf{p}_i(\mathbf{u}) \| \mathbf{p}_i(0)\right) = \sum_{j:\overline{p}_{ij} \neq 0} p_{ij}(\mathbf{u}) \log \frac{p_{ij}(\mathbf{u})}{p_{ij}(0)} \tag{4}$$

From the properties of KL divergence it follows that $r(i, \mathbf{u}) \geq 0$, and $r(i, \mathbf{u}) = 0$ iff $\mathbf{u} = 0$. Substituting (2) in (4) and simplifying, the control cost becomes

$$r(i, \mathbf{u}) = \sum_j p_{ij}(\mathbf{u}) u_j \tag{5}$$

This has an interesting interpretation. The Markov chain likes to behave according to $\overline{P}$ but can be paid to behave according to $P(\mathbf{u})$. Before each transition the controller specifies the price $u_j$ it is willing to pay (or collect, if $u_j < 0$) for every possible next state $j$. When the actual transition occurs, say to state $k$, the controller pays the price $u_k$ it promised. Then $r(i, \mathbf{u})$ is the price the controller expects to pay before observing the transition.

Coming back to the MDP construction, we allow an arbitrary state cost $q(i) \geq 0$ in addition to the above control cost:

$$\ell(i, \mathbf{u}) = q(i) + r(i, \mathbf{u}) \tag{6}$$

We require $q(i) = 0$ for absorbing states $i \in \mathcal{A}$ so that the process can continue indefinitely without incurring extra costs. Substituting (5, 6) in (1), the Bellman equation for our MDP is

$$v(i) = \min_{\mathbf{u} \in \mathcal{U}(i)} \left\{ q(i) + \sum_j \overline{p}_{ij} \exp(u_j)(u_j + v(j)) \right\} \tag{7}$$

We can now exploit the benefits of this unusual construction. The minimization in (7) subject to the constraint (3) can be performed in closed form using Lagrange multipliers, as follows. For each $i$ define the Lagrangian

$$\mathcal{L}(\mathbf{u}, \lambda_i) = \sum_j \overline{p}_{ij} \exp(u_j)(u_j + v(j)) + \lambda_i \left( \sum_j \overline{p}_{ij} \exp(u_j) - 1 \right) \tag{8}$$

The necessary condition for an extremum with respect to $u_j$ is

$$0 = \frac{\partial \mathcal{L}}{\partial u_j} = \overline{p}_{ij} \exp(u_j)(u_j + v(j) + \lambda_i + 1) \tag{9}$$

When $\overline{p}_{ij} \neq 0$ the only solution is

$$u_j^*(i) = -v(j) - \lambda_i - 1 \tag{10}$$

Taking another derivative yields

$$\left.\frac{\partial^2 \mathcal{L}}{\partial u_j \partial u_j}\right|_{u_j = u_j^*(i)} = \overline{p}_{ij} \exp\left(u_j^*(i)\right) > 0 \tag{11}$$

and therefore (10) is a minimum. The Lagrange multiplier $\lambda_i$ can be found by applying the constraint (3) to the optimal control (10). The result is

$$\lambda_i = \log\left(\sum_j \overline{p}_{ij} \exp\left(-v\left(j\right)\right)\right) - 1 \tag{12}$$

and therefore the optimal control law is

$$u_j^*(i) = -v(j) - \log\left(\sum_k \overline{p}_{ik} \exp\left(-v\left(k\right)\right)\right) \tag{13}$$

Thus we have expressed the optimal control law in closed form given the optimal value function. Note that the only influence of the current state $i$ is through the second term, which serves to normalize the transition probability distribution $\mathbf{p}_i(\mathbf{u}^*)$ and is identical for all next states $j$. Thus the optimal controller is a high-level controller: it tells the Markov chain to go to good states without specifying how to get there. The details of the trajectory emerge from the interaction of this controller and the uncontrolled stochastic dynamics. In particular, the optimally-controlled transition probabilities are

$$p_{ij}\left(\mathbf{u}^*(i)\right) = \frac{\overline{p}_{ij} \exp\left(-v\left(j\right)\right)}{\sum_k \overline{p}_{ik} \exp\left(-v\left(k\right)\right)} \tag{14}$$

These probabilities are proportional to the product of two terms: the uncontrolled transition probabilities $\overline{p}_{ij}$ which do not depend on the costs or values, and the (exponentiated) next-state values $v(j)$ which do not depend on the current state. In the special case $\overline{p}_{ij} = const_i$ the transition probabilities (14) correspond to a Gibbs distribution where the optimal value function plays the role of an energy function.

Substituting the optimal control (13) in the Bellman equation (7) and dropping the min operator,

$$
\begin{aligned}
v(i) &= q(i) + \sum_j p_{ij}\left(\mathbf{u}^*(i)\right)\left(u_j^*(i) + v(j)\right) \\
&= q(i) + \sum_j p_{ij}\left(\mathbf{u}^*(i)\right)\left(-\lambda_i - 1\right) \\
&= q(i) - \lambda_i - 1 \\
&= q(i) - \log\left(\sum_j \overline{p}_{ij} \exp\left(-v\left(j\right)\right)\right)
\end{aligned} \tag{15}
$$

Rearranging terms and exponentiating both sides of (15) yields

$$\exp\left(-v\left(i\right)\right) = \exp\left(-q\left(i\right)\right) \sum_j \overline{p}_{ij} \exp\left(-v\left(j\right)\right) \tag{16}$$

We now introduce the exponential transformation

$$z(i) = \exp\left(-v\left(i\right)\right) \tag{17}$$

which makes the minimized Bellman equation linear:

$$z(i) = \exp\left(-q\left(i\right)\right) \sum_j \overline{p}_{ij} z(j) \tag{18}$$

Defining the vector $\mathbf{z}$ with elements $z(i)$, and the diagonal matrix $G$ with elements $\exp\left(-q\left(i\right)\right)$ along its main diagonal, (18) becomes

$$\mathbf{z} = G\overline{P}\mathbf{z} \tag{19}$$

Thus our class of optimal control problems has been reduced to a linear eigenvalue problem.

## 2.1 Iterative solution and convergence analysis

From (19) it follows that $\mathbf{z}$ is an eigenvector of $G\overline{P}$ with eigenvalue 1. Furthermore $z(i) > 0$ for all $i \in \mathcal{S}$ and $z(i) = 1$ for $i \in \mathcal{A}$. Is there a vector $\mathbf{z}$ with these properties and is it unique? The answer to both questions is affirmative, because the Bellman equation has a unique solution and $v$ is a solution to the Bellman equation iff $z = \exp(-v)$ is an admissible solution to (19). The only remaining question then is how to find the unique solution $\mathbf{z}$. The obvious iterative method is

$$\mathbf{z}_{k+1} = G\overline{P}\mathbf{z}_k, \qquad \mathbf{z}_0 = \mathbf{1} \tag{20}$$

This iteration always converges to the unique solution, for the following reasons. A stochastic matrix $\overline{P}$ has spectral radius 1. Multiplication by $G$ scales down some of the rows of $\overline{P}$, therefore $G\overline{P}$ has spectral radius at most 1. But we are guaranteed than an eigenvector $\mathbf{z}$ with eigenvalue 1 exists, therefore $G\overline{P}$ has spectral radius 1 and $\mathbf{z}$ is a largest eigenvector. Iteration (20) is equivalent to the power method (without the rescaling which is unnecessary here) so it converges to a largest eigenvector. The additional constraints on $\mathbf{z}$ are clearly satisfied at all stages of the iteration. In particular, for $i \in \mathcal{A}$ the $i$-th row of $G\overline{P}$ has elements $\delta_i^j$, and so the $i$-th element of $\mathbf{z}_k$ remains equal to 1 for all $k$.

We now analyze the rate of convergence. Let $m = |\mathcal{A}|$ and $n = |\mathcal{S}|$. The states can be permuted so that $G\overline{P}$ is in canonical form:

$$G\overline{P} = \begin{bmatrix} T_1 & T_2 \\ 0 & I \end{bmatrix} \tag{21}$$

where the absorbing states are last, $T_1$ is $(n-m)$ by $(n-m)$, and $T_2$ is $(n-m)$ by $m$. The reason we have the identity matrix in the lower-right corner, despite multiplication by $G$, is that $q(i) = 0$ for $i \in \mathcal{A}$. From (21) we have

$$\left(G\overline{P}\right)^k = \begin{bmatrix} T_1^k & \left(T_1^{k-1} + \cdots + T_1 + I\right) T_2 \\ 0 & I \end{bmatrix} = \begin{bmatrix} T_1^k & \left(I - T_1^k\right)\left(I - T_1\right)^{-1} T_2 \\ 0 & I \end{bmatrix} \tag{22}$$

A stochastic matrix $\overline{P}$ with $m$ absorbing states has $m$ eigenvalues 1, and all other eigenvalues are smaller than 1 in absolute value. Since the diagonal elements of $G$ are no greater than 1, all eigenvalues of $T_1$ are smaller than 1 and so $\lim_{k \to \infty} T_1^k = 0$. Therefore iteration (20) converges exponentially as $\gamma^k$ where $\gamma < 1$ is the largest eigenvalue of $T_1$. Faster convergence is obtained for smaller $\gamma$. The factors that can make $\gamma$ small are: (i) large state costs $q(i)$ resulting in small terms $\exp(-q(i))$ along the diagonal of $G$; (ii) small transition probabilities among non-absorbing states (and large transition probabilities from non-absorbing to absorbing states). Convergence is independent of problem size because $\gamma$ has no reason to increase as the dimensionality of $T_1$ increases. Indeed numerical simulations on randomly generated MDPs have shown that problem size does not systematically affect the number of iterations needed to reach a given convergence criterion. Thus the average running time scales linearly with the number of non-zero elements in $\overline{P}$.

## 2.2 Alternative problem formulations

While the focus of this paper is on infinite-horizon total-cost problems with absorbing states, we have obtained similar results for all other problem formulations commonly used in Reinforcement Learning. Here we summarize these results. In finite-horizon problems equation (19) becomes

$$\mathbf{z}(t) = G(t)\,\overline{P}(t)\,\mathbf{z}(t+1) \tag{23}$$

where $\mathbf{z}(t_{\text{final}})$ is initialized from a given final cost function. In infinite-horizon average-cost-per-stage problems equation (19) becomes

$$\gamma\mathbf{z} = G\overline{P}\mathbf{z} \tag{24}$$

where $\gamma$ is the largest eigenvalue of $G\overline{P}$, $\mathbf{z}$ is a differential value function, and the average cost-per-stage turns out to be $-\log(\gamma)$. In infinite-horizon discounted-cost problems equation (19) becomes

$$\mathbf{z} = G\overline{P}\mathbf{z}^\alpha \tag{25}$$

where $\alpha < 1$ is the discount factor and $\mathbf{z}^\alpha$ is defined element-wise. Even though the latter equation is nonlinear, we have observed that the analog of iteration (20) still converges rapidly.

Fig 1A

| 0 | 3 | 4 | 6 | 8 | 10 | 12 |
|---|---|---|---|---|----|----|
| 2 | 3 | 4 | 6 | 9 | 11 | 12 |
|   |   |   |   |   | 11 | 13 |
| 22 | 20 | 18 | 16 | 14 | 14 | 13 |
| 22 | 21 |   |   |   |   |   |
| 23 | 23 | 24 | 26 | 28 | 30 | 31 |
| 25 | 25 | 25 | 26 | 28 | 30 | 31 |

Fig 1B

| 0 | 1 | 2 | 3 | 4 | 5 | 6 |
|---|---|---|---|---|---|---|
| 1 | 1 | 2 | 3 | 4 | 5 | 6 |
|   |   |   |   |   | 5 | 6 |
| 10 | 9 | 8 | 7 | 6 | 6 | 6 |
| 10 | 9 |   |   |   |   |   |
| 10 | 10 | 10 | 11 | 12 | 13 | 14 |
| 11 | 11 | 11 | 11 | 12 | 13 | 14 |

## 3 Shortest paths as an eigenvalue problem

Suppose the state space $\mathcal{S}$ of our MDP corresponds to the vertex set of a directed graph, and let $D$ be the graph adjacency matrix whose element $d_{ij}$ indicates the presence $(d_{ij} = 1)$ or absence $(d_{ij} = 0)$ of a directed edge from vertex $i$ to vertex $j$. Let $\mathcal{A} \subseteq \mathcal{S}$ be a non-empty set of destination vertices. Our goal is to find the length $s(i)$ of the shortest path from every $i \in \mathcal{S}$ to some vertex in $\mathcal{A}$. For $i \in \mathcal{A}$ we have $s(i) = 0$ and $d_{ij} = \delta_i^j$.

We now show how the shortest path lengths $s(i)$ can be obtained from our MDP. Define the elements of the stochastic matrix $\overline{P}$ as

$$\overline{p}_{ij} = \frac{d_{ij}}{\sum_k d_{ik}} \tag{26}$$

corresponding to a random walk on the graph. Next choose $\rho > 0$ and define the state costs

$$q_\rho(i) = \rho \text{ when } i \notin \mathcal{A}, \qquad q_\rho(i) = 0 \text{ when } i \in \mathcal{A} \tag{27}$$

This cost model means that we pay a price $\rho$ whenever the current state is not in $\mathcal{A}$. Let $v_\rho(i)$ denote the optimal value function for the MDP defined by (26, 27). If the control costs were $0$ then the shortest paths would simply be $s(i) = \frac{1}{\rho} v_\rho(i)$. Here the control costs are not $0$, however they are bounded. This can be shown using

$$p_{ij}(\mathbf{u}) = \overline{p}_{ij} \exp(u_j) \le 1 \tag{28}$$

which implies that for $\overline{p}_{ij} \ne 0$ we have $u_j \le -\log(\overline{p}_{ij})$. Since $r(i, \mathbf{u})$ is a convex combination of the elements of $\mathbf{u}$, the following bound holds:

$$r(i, \mathbf{u}) \le \max_j(u_j) \le -\log\left(\min_{j:\overline{p}_{ij} \ne 0}(\overline{p}_{ij})\right) \tag{29}$$

The control costs are bounded and we are free to choose $\rho$ arbitrarily large, so we can make the state costs dominate the optimal value function. This yields the following result:

$$s(i) = \lim_{\rho \to \infty} \frac{v_\rho(i)}{\rho} \tag{30}$$

Thus we have reduced the shortest path problem to an eigenvalue problem. In spectral graph theory many problems have previously been related to eigenvalues of the graph Laplacian [4], but the shortest path problem was not among them until now. Currently the most widely used algorithm is Dijkstra's algorithm. In sparse graphs its running time is $O(n \log(n))$. In contrast, algorithms for finding largest eigenpairs have running time $O(n)$ for sparse matrices.

Of course (30) involves a limit and so we cannot obtain the exact shortest paths by solving a single eigenvalue problem. However we can obtain a good approximation by setting $\rho$ large enough – but not too large because $\exp(-\rho)$ may become numerically indistinguishable from $0$. **Fig 1** illustrates the solution obtained from (30) and rounded down to the nearest integer, for $\rho = 1$ in **1A** and $\rho = 50$ in **1B**. Transitions are allowed to all neighbors. The result in **1B** matches the exact shortest paths. Although the solution for $\rho = 1$ is numerically larger, it is basically a scaled-up version of the correct solution. Indeed the $R^2$ between the two solutions before rounding was $0.997$.

# 4 Approximating discrete MDPs via continuous embedding

In the previous section we replaced the shortest path problem with a continuous MDP and obtained an excellent approximation. Here we obtain approximations of similar quality in more general settings, using an approach reminiscent of LP-relaxation in integer programming. As in LP-relaxation, theoretical results are hard to derive but empirically the method works well.

We construct an embedding which associates the controls in the discrete MDP with specific control vectors of a continuous MDP, making sure that for these control vectors the continuous MDP has the same costs and transition probabilities as the discrete MDP. This turns out to be possible under mild and reasonable assumptions, as follows. Consider a discrete MDP with transition probabilities and costs denoted $\widetilde{p}$ and $\widetilde{\ell}$. Define the matrix $B(i)$ of all controlled transition probabilities from state $i$. This matrix has elements

$$b_{aj}(i) = \widetilde{p}_{ij}(a), \qquad a \in \mathcal{U}(i) \tag{31}$$

We need two assumptions to guarantee the existence of an exact embedding: for all $i \in \mathcal{S}$ the matrix $B(i)$ must have full row-rank, and if any element of $B(i)$ is 0 then the entire column must be 0. If the latter assumption does not hold, we can replace the problematic 0 elements of $B(i)$ with a small $\epsilon$ and renormalize. Let $\mathcal{N}(i)$ denote the set of possible next states, i.e. states $j$ for which $\widetilde{p}_{ij}(a) > 0$ for any/all $a \in \mathcal{U}(i)$. Remove the zero-columns of $B(i)$ and restrict $j \in \mathcal{N}(i)$.

The first step in the construction is to compute the real-valued control vectors $\mathbf{u}^a$ corresponding to the discrete actions $a$. This is accomplished by matching the transition probabilities of the discrete and continuous MDPs:

$$\overline{p}_{ij} \exp\left(u_j^a\right) = \widetilde{p}_{ij}(a), \qquad \forall\, i \in \mathcal{S},\; j \in \mathcal{N}(i),\; a \in \mathcal{U}(i) \tag{32}$$

These constraints are satisfied iff the elements of the vector $\mathbf{u}^a$ are

$$u_j^a = \log\left(\widetilde{p}_{ij}(a)\right) - \log\left(\overline{p}_{ij}\right) \tag{33}$$

The second step is to compute the uncontrolled transition probabilities $\overline{p}_{ij}$ and state costs $q(i)$ in the continuous MDP so as to match the costs in the discrete MDP. This yields the set of constraints

$$q(i) + r(i, \mathbf{u}^a) = \widetilde{\ell}(i, a), \qquad \forall\, i \in \mathcal{S},\; a \in \mathcal{U}(i) \tag{34}$$

For the control vector given by (33) the KL-divergence cost is

$$r(i, \mathbf{u}^a) = \sum\nolimits_j \overline{p}_{ij} \exp\left(u_j^a\right) u_j^a = h(i, a) - \sum\nolimits_j \widetilde{p}_{ij}(a) \log\left(\overline{p}_{ij}\right) \tag{35}$$

where $h(i, a)$ is the entropy of the transition probability distribution in the discrete MDP:

$$h(i, a) = \sum\nolimits_j \widetilde{p}_{ij}(a) \log\left(\widetilde{p}_{ij}(a)\right) \tag{36}$$

The constraints (34) are then equivalent to

$$q(i) - \sum\nolimits_j b_{aj}(i) \log\left(\overline{p}_{ij}\right) = \widetilde{\ell}(i, a) - h(i, a) \tag{37}$$

Define the vector $\mathbf{y}(i)$ with elements $\widetilde{\ell}(i, a) - h(i, a)$, and the vector $\mathbf{x}(i)$ with elements $\log\left(\overline{p}_{ij}\right)$. The dimensionality of $\mathbf{y}(i)$ is $|\mathcal{U}(i)|$ while the dimensionality of $\mathbf{x}(i)$ is $|\mathcal{N}(i)| \geq |\mathcal{U}(i)|$. The latter inequality follows from the assumption that $B(i)$ has full row-rank. Suppressing the dependence on the current state $i$, the constraints (34) can be written in matrix notation as

$$q\mathbf{1} - B\mathbf{x} = \mathbf{y} \tag{38}$$

Since the probabilities $\overline{p}_{ij}$ must sum up to 1, the vector $\mathbf{x}$ must satisfy the additional constraint

$$\sum\nolimits_j \exp(x_j) = 1 \tag{39}$$

We are given $B, \mathbf{y}$ and need to compute $q, \mathbf{x}$ satisfying (38, 39). Let $\widehat{\mathbf{x}}$ be any vector such that $-B\widehat{\mathbf{x}} = \mathbf{y}$, for example $\widehat{\mathbf{x}} = -B^\dagger \mathbf{y}$ where $^\dagger$ denotes the Moore-Penrose pseudoinverse. Since $B$ is a stochastic matrix we have $B\mathbf{1} = \mathbf{1}$, and so

$$q\mathbf{1} - B\left(\widehat{\mathbf{x}} + q\mathbf{1}\right) = -B\widehat{\mathbf{x}} = \mathbf{y} \tag{40}$$

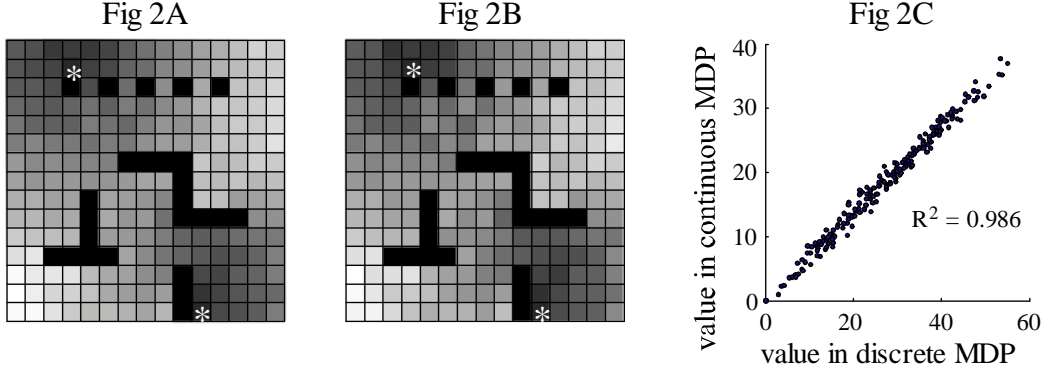

Fig 2A              Fig 2B              Fig 2C

Therefore $\mathbf{x} = \widehat{\mathbf{x}} + q\mathbf{1}$ satisfies (38) for all $q$, and we can adjust $q$ to also satisfy (39), namely

$$q = -\log\left(\sum_j \exp\left(\widehat{x}_j\right)\right) \tag{41}$$

This completes the embedding. If the above $q$ turns out to be negative, we can either choose another $\widehat{\mathbf{x}}$ by adding an element from the null-space of $B$, or scale all costs $\widetilde{\ell}(i, a)$ by a positive constant. Such scaling does not affect the optimal control law for the discrete MDP, but it makes the elements of $-B^{\dagger}\mathbf{y}$ more negative and thus $q$ becomes more positive.

We now illustrate this construction with the example in **Fig 2**. The grid world has a number of obstacles (black squares) and two absorbing states (white stars). The possible next states are the immediate neighbors including the current state. Thus $|\mathcal{N}(i)|$ is at most 9. The discrete MDP has $|\mathcal{N}(i)| - 1$ actions corresponding to stochastic transitions to each of the neighbors. For each action, the transition probability to the "desired" state is $0.8$ and the remaining $0.2$ is equally distributed among the other states. The costs $\widetilde{\ell}(i, a)$ are random numbers between 1 and 10 – which is why the optimal value function shown in grayscale appears irregular. **Fig 2A** shows the optimal value function for the discrete MDP. **Fig 2B** shows the optimal value function for the corresponding continuous MDP. The scatterplot in **Fig 2C** shows the optimal values in the discrete and continuous MDP (each dot is a state). The values in the continuous MDP are numerically smaller – which is to be expected since the control space is larger. Nevertheless, the correlation between the optimal values in the discrete and continuous MDPs is excellent. We have observed similar performance in a number of randomly-generated problems.

## 5 Z-learning

So far we assumed that a model of the continuous MDP is available. We now turn to stochastic approximations of the optimal value function which can be used when a model is not available. All we have access to are samples $(i_k, j_k, q_k)$ where $i_k$ is the current state, $j_k$ is the next state, $q_k$ is the state cost incurred at $i_k$, and $k$ is the sample number. Equation (18) can be rewritten as

$$z(i) = \exp\left(-q(i)\right)\sum_j \overline{p}_{ij}z(j) = \exp\left(-q(i)\right)E_{\overline{P}}\left[z(j)\right] \tag{42}$$

This suggests an obvious stochastic approximation $\widehat{z}$ to the function $z$, namely

$$\widehat{z}(i_k) \leftarrow (1 - \alpha_k)\,\widehat{z}(i_k) + \alpha_k \exp\left(-q_k\right)\widehat{z}(j_k) \tag{43}$$

where the sequence of learning rates $\alpha_k$ is appropriately decreased as $k$ increases. The approximation to $v(i)$ is simply $-\log\left(\widehat{z}(i)\right)$. We will call this algorithm Z-learning.

Let us now compare (43) to the Q-learning algorithm applicable to discrete MDPs. Here we have samples $(i_k, j_k, \ell_k, u_k)$. The difference is that $\ell_k$ is now a total cost rather than a state cost, and we have a control $u_k$ generated by some control policy. The update equation for Q-learning is

$$\widehat{Q}(i_k, u_k) \leftarrow (1 - \alpha_k)\,\widehat{Q}(i_k, u_k) + \alpha_k \min_{u' \in \mathcal{U}(j_k)}\left(\ell_k + \widehat{Q}(j_k, u')\right) \tag{44}$$

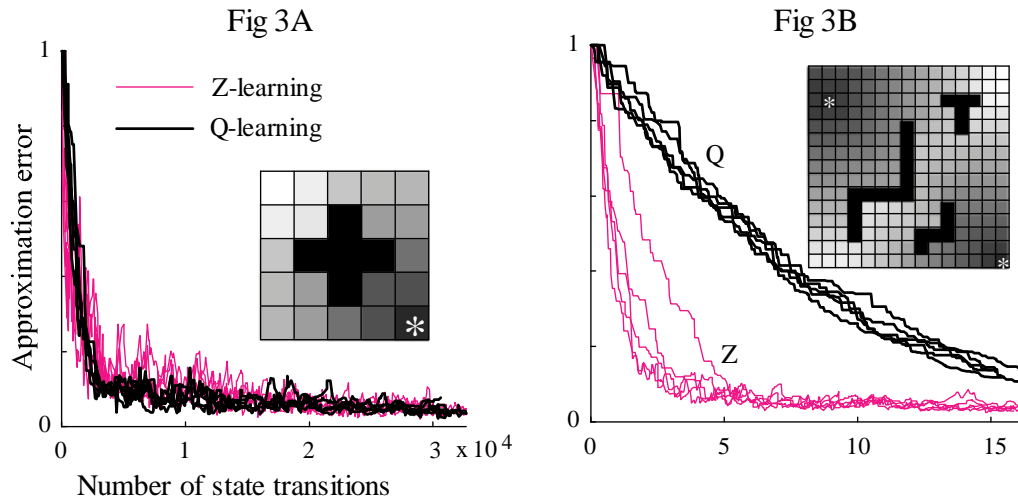

Fig 3A · Fig 3B

To compare the two algorithms, we first constructed continuous MDPs with $q(i) = 1$ and transitions to the immediate neighbors in the grid worlds shown in **Fig 3**. For each state we found the optimal transition probabilities (14). We then constructed a discrete MDP which had one action (per state) that caused the same transition probabilities, and the corresponding cost was the same as in the continuous MDP. We then added $|\mathcal{N}(i)| - 1$ other actions by permuting the transition probabilities. Thus the discrete and continuous MDPs were guaranteed to have identical optimal value functions. Note that the goal here is no longer to approximate discrete with continuous MDPs, but to construct pairs of problems with identical solutions allowing fair comparison of Z-learning and Q-learning.

We run both algorithms with the same random policy. The learning rates decayed as $\alpha_k = c/(c + t(k))$ where the constant $c$ was optimized separately for each algorithm and $t(k)$ is the run to which sample $k$ belongs. When the MDP reaches an absorbing state a new run is started from a random initial state. The approximation error plotted in **Fig 3** is defined as

$$\frac{\max_i |v(i) - \widehat{v}(i)|}{\max_i v(i)} \tag{45}$$

and is computed at the end of each run. For small problems (**Fig 3A**) the two algorithms had identical convergence, however for larger problems (**Fig 3B**) the new Z-learning algorithm was clearly faster. This is not surprising: even though Z-learning is as model-free as Q-learning, it benefits from the analytical developments in this paper and in particular it does not need a maximization operator or state-action values. The performance of Q-learning can be improved by using a non-random (say $\epsilon$-greedy) policy. If we combine Z-learning with importance sampling, the performance of Z-learning can also be improved by using such a policy.

## 6 Summary

We introduced a new class of MDPs which have a number of remarkable properties, can be solved efficiently, and yield accurate approximations to traditional MDPs. In general, no single approach is likely to be a magic wand which simplifies all optimal control problems. Nevertheless the results so far are very encouraging. While the limitations remain to be clarified, our approach appears to have great potential and should be thoroughly investigated.

## References

[1] B. Scholkopf and A. Smola, Learning with kernels. MIT Press (2002)

[2] S. Boyd and L. Vandenberghe, Convex optimization. Cambridge University Press (2004)

[3] D. Bertsekas, Dynamic programming and optimal control (2nd ed). Athena Scientific (2000)

[4] F. Chung, Spectral graph theory. CMBS Regional Conference Series in Mathematics (1997)
